# Qualitative structure from motion

**Daphna Weinshall**
Center for Biological Information Processing
MIT, E25-201, Cambridge MA 02139

## Abstract

Exact structure from motion is an ill-posed computation and therefore very sensitive to noise. In this work I describe how a qualitative shape representation, based on the sign of the Gaussian curvature, can be computed directly from motion disparities, without the computation of an exact depth map or the directions of surface normals. I show that humans can judge the curvature sense of three points undergoing 3D motion from two, three and four views with success rate significantly above chance. A simple RBF net has been trained to perform the same task.

## 1  INTRODUCTION

When a scene is recorded from two or more different positions in space, e.g. by a moving camera, objects are projected into disparate locations in each image. This disparity can be used to recover the three-dimensional structure of objects that is lost in the projection process. The computation of structure requires knowledge of the 3D motion parameters. Although these parameters can themselves be computed from the disparities, their computation presents a difficult problem that is mathematically ill-posed: small perturbations (or errors) in the data may cause large changes in the solution [9]. This brittleness, or sensitivity to noise, is a major factor limiting the applicability of a number of structure from motion algorithms in practical situations (Ullman, 1983).

The problem of brittleness of the structure from motion algorithms that use the minimal possible information may be attacked through two different approaches. One involves using more data, either in the space domain (more corresponding points in each image frame, Bruss & Horn, 1981), or in the time domain (more frames,

Ullman, 1984). The other approach is to look for, instead of a general quantitative solution, a qualitative one that would still meet the main requirements of the task for which the computation is performed (e.g., object representation or navigation). This approach has been applied to navigation (e.g., Nelson & Aloimonos, 1988) and object recognition (e.g., Koenderink & van Doorn, 1976; Weinshall, 1989).

Under perspective projection, the knowledge of the positions of 7 corresponding points in two successive frames is the theoretical lower limit of information necessary to compute the 3D structure of an object that undergoes a general motion (Tsai & Huang, 1984). As mentioned above, acceptable performance of structure from motion algorithms on real, noisy images requires that a larger number of corresponding points be used. In contrast, the human visual system can extract 3D motion information using as few as 3 points in each of the two frames (Borjesson & von Hofsten, 1973). To what extent can object shape be recovered from such impoverished data? I have investigated this question experimentally (by studying the performance of human subjects) and theoretically (by analyzing the information available in the three-point moving stimuli).

## 2   THEORETICAL SHORTCUTS

The goal of the structure from motion computation is to obtain the depth map of a moving object: the value of the depth coordinate at each point in the 2D image of the object. The depth map can be used subsequently to build a representation of the object, e.g., for purposes of recognition. One possible object representation is the description of an object as a collection of generic parts, where each part is described by a few parameters. Taking the qualitative approach to vision described in the introduction, the necessity of having a complete depth map for building useful generic representations can be questioned. Indeed, one such representation, a map of the sign of the Gaussian curvature of the object's surface, can be computed directly (and, possibly, more reliably) from motion disparities. The knowledge of the sign of the Gaussian curvature of the surface allows the classification of surface patches as elliptic (convex/concave), hyperbolic (saddle point), cylindrical, or planar. Furthermore, the boundaries between adjacent generic parts are located along lines of zero curvature (parabolic lines).

The basic result that allows the computation of the sign of the Gaussian curvature directly from motion disparities is the following theorem (see Weinshall, 1989 for details):

**Theorem 1** *Let FOE denote the Focus Of Expansion – the location in the image towards (or away from) which the motion is directed.*

*Pick three collinear points in one image and observe the pattern they form in a subsequent image.*

*The sign of the curvature of these three points in the second image relative to the FOE is the same as the sign of the normal curvature of the 3D curve defined by these three points.*

The sign of the Gaussian curvature at a given point can be found without knowing the direction of the normal to the surface, by computing the curvature sign of point

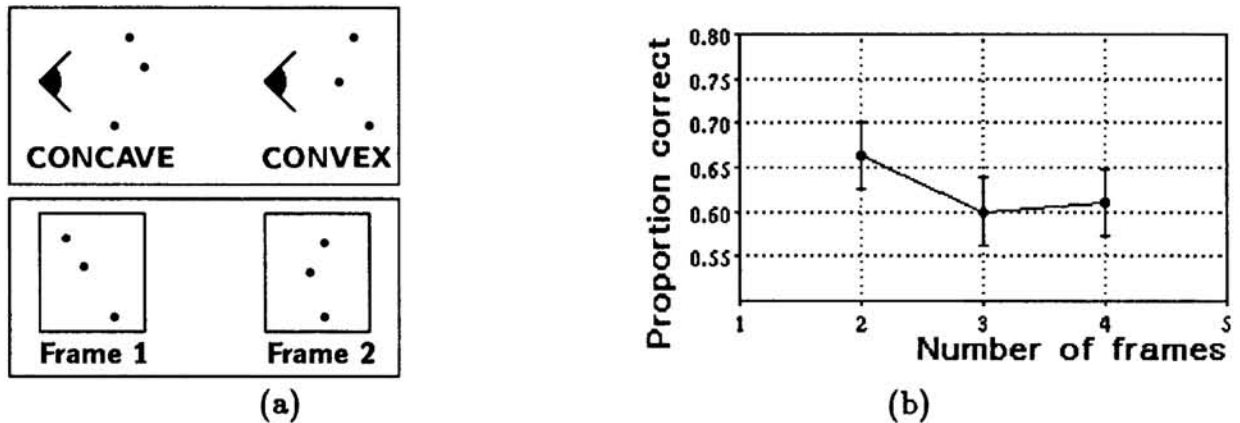

Figure 1: Experiment 1: perception of curvature from three points in 3D translation. *(a)* Four naive subjects were shown two, three or four snapshots of the motion sequence. The subjects did not perceive the motion as translation. The total extent and the speed of the motion were identical in each condition. The three points were always collinear in the first frame. The back and forth motion sequence was repeated eight times, after which the subjects were required to decide on the sign of the curvature (see text). The mean performance, 62%, differed significantly from chance ($t = 5.55$, $p < 0.0001$). Furthermore, all subjects but one performed significantly above chance. *(b)* The effect of the number of frames was not significant ($\chi^2 = 1.72$, $p = 0.42$). Bars show ±1 standard error of the mean.

triads in all directions around the point. The sign of the Gaussian curvature is determined by the number of sign reversals of the triad curvatures encountered around the given point. The exact location of the FOE is therefore not important.

The sign operator described above has biological appeal, since the visual system can compute the deviation of three points from a straight line with precision in the hyperacuity range (that is, by an order of magnitude more accurately than allowed by the distance between adjacent photoreceptors in the retina). In addition, this feature must be important to the visual system, since it appears to be detected preattentively (in parallel over the entire visual field; see Fahle, 1990).

It is difficult to determine whether the visual system uses such a qualitative strategy to characterize shape features, since it is possible that complete structure is first recovered, from which the sign of the Gaussian curvature is then computed. In the following experiments I present subjects with impoverished data that is insufficient for exact structure from motion (3 points in 2 frames). If subjects can perform the task, they have to use some strategy different from exact depth recovery.

## 3   EXPERIMENT 1

In the first experiment four subjects were presented with 120 moving rigid configurations of three points. The number of distinct frames per configuration varied from 2 to 4. The motion was translation only. Subjects had to judge whether the three points were in a convex or a concave configuration, namely, whether the broken 3D

line formed by the points was bent towards or away from the subject (figure 1a). The middle point was almost never the closest or the farthest one, so that relative depth was not sufficient for solving the problem. With only two-frame the stimulus was ambiguous in that there was an infinity of rigid convex and concave 3D configurations of three points that could have given rise to the images presented. For these stimuli the correct answer is meaningless, and one important question is whether this inherent ambiguity affects the subjects' performance (as compared to their performance with 3 and 4 frames).

The subjects' performance in this experiment was significantly better than chance (figure 1b). The subjects were able to recover partial information on the shape of the stimulus even with 2 frames, despite the theoretical impossibility of a full structure from motion computation[1]. Moreover, the number of frames presented in each trial had no significant effect on the error rate: the subjects performed just as well in the 2 frame trials as in the 3 and 4 frame trials (figure 1b). Had the subjects relied on the exact computation of structure from motion, one would expect a better performance with more frames (Ullman, 1984; Hildreth et al., 1989).

One possible account (reconstructional) of this result is that subjects realized that the motion of the stimuli consisted of pure 3D translation. Three points in two frames are in principle sufficient to verify that the motion is translational and to compute the translation parameters. The next experiment renders this account implausible by demonstrating that the subjects perform as well when the stimuli undergo general motion that includes rotation as well as translation.

Another possible (geometrical) account is that the human visual system incorporates the geometrical knowledge expressed by theorem 1, and uses this knowledge in ambiguous cases to select the more plausible answer. However, theorem 1 does not address the ambiguity of the stimulus that stems from the dependency of the result on the location of the Focus Of Expansion. If indeed some knowledge of this theorem is used in performing this task, the ambiguity has to be resolved by "guessing" the location of the FOE. The strategy consistent with human performance in the first experiment is assuming that the FOE lies in the general direction towards which the points in the image are moving. The next experiment is designed to check the use of this heuristic.

## 4  EXPERIMENT 2

This experiment was designed to clarify which of the two proposed explanations to the subjects' good performance in experiment 1 with only 2 frames is more plausible.

First, to eliminate completely the cue to exact depth in a translational motion, the stimuli in experiment 2 underwent rotation as well as translation. The 3D motion was set up in such a manner that the projected 2D optical flow could not be interpreted as resulting from pure translational motion.

Second, if subjects do use an implicit knowledge of theorem 1, the accuracy of their performance should depend on the correctness of the heuristic used to estimate

the location of the FOE as discussed in the previous section. This heuristic yields incorrect results for many instances of general 3D motion. In experiment 2, two types of 3-point 2-frame motion were used: one in which the estimation of the FOE using the above heuristic is correct, and one in which this estimation is wrong. If subjects rely on an implicit knowledge of theorem 1, their judgement should be mostly correct for the first type of motion, and mostly incorrect for the second type.

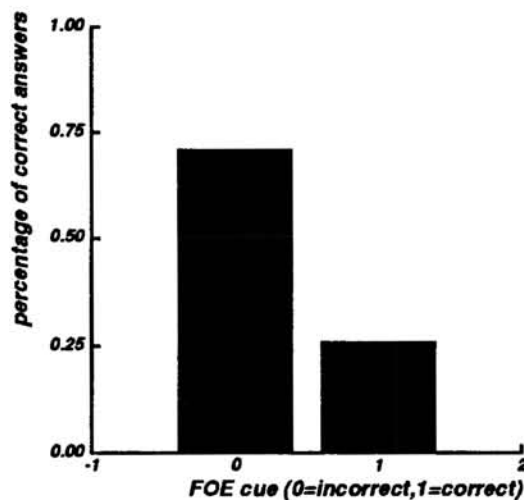

Figure 2: Experiment 2: three points in general motion. The same four subjects as in experiment 1 were shown two-frame sequences of back and forth motion that included 3D translation and rotation. The mean performance when the FOE heuristic (see text) was correct, 71%, was significantly above chance ($t = 5.71$, $p < 0.0001$). In comparison, the mean performance when the FOE heuristic was misleading, 26%, was significantly below chance ($t = -4.90$, $p < 0.0001$). The degree to which the motion could be mistakenly interpreted as pure translation was uncorrelated with performance ($r = 0.04$, $F(1, 318) < 1$). The performance in experiment 2 was similar to that in experiment 1 (the difference was not significant $\chi^2 < 1$). In other words, the performance was as good under general motion as under pure translation.

Figure 2a describes the results of experiment 2. As in the first experiment, the subjects performed significantly above chance when the FOE estimation heuristic was correct. When the heuristic was misleading, they were as likely to be wrong as they were likely to be right in the correct heuristic condition. As predicted by the geometrical explanation to the first experiment, seeing general motion instead of pure translation did not seem to affect the performance.

## 5    LEARNING WITH A NEURAL NETWORK

Computation of qualitative structure from motion, outlined in section 2, can be supported by a biologically plausible architecture based on the application of a three-point hyperacuity operator, in parallel, in different directions around each point and over the entire visual field. Such a computation is particularly suitable to implementation by an artificial neural network. I have trained a Radial Basis Function (RBF) network (Moody & Darken, 1989; Poggio & Girosi, 1990) to

identify the sign of Gaussian curvature of three moving points (represented by a coordinate vector of length 6). After a supervised learning phase in which the network was trained to produce the correct sign given examples of motion sequences, it consistently achieved a substantial success rate on novel inputs, for a wide range of parameters. Figure 3 shows the success rate (the percentage of correct answers) plotted against the number of examples used in the training phase.

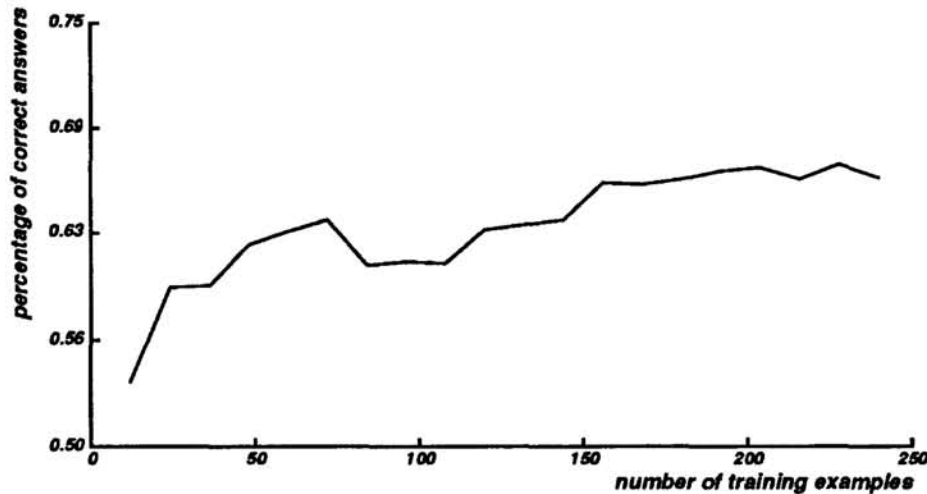

Figure 3: The correct performance rate of the RBF implementation vs. the number of examples in the training set.

## 6   SUMMARY

I have presented a qualitative approach to the problem of recovering object structure from motion information and discussed some of its computational, psychophysical and implementational aspects. The computation of qualitative shape, as represented by the sign of the Gaussian curvature, can be performed by a field of simple operators, in parallel over the entire image. The performance of a qualitative shape detection module, implemented by an artificial neural network, appears to be similar to the performance of human subjects in an identical task.

### Acknowledgements

I thank H. Bülthoff, N. Cornelius, M. Dornay, S. Edelman, M. Fahle, S. Kirkpatrick, M. Ross and A. Shashua for their help. This research was done partly in the MIT AI Laboratory. It was supported by a Fairchild postdoctoral fellowship, and in part by grants from the office of Naval Research (N00014-88-k-0164), from the National Science Foundation (IRI-8719394 and IRI-8657824), and a gift from the James S. McDonnell Foundation to Professor Ellen Hildreth.

## Footnotes

[1]I should note that all the subjects were surprised by their good performance. They felt that the stimulus was ambiguous and that they were mostly guessing.

### References

[1] E. Borjesson and C. von Hofsten. Visual perception of motion in depth: ap-

plication of a vector model to three-dot motion patterns. *Perception and Psychophysics*, 13:169–179, 1973.

[2] A. Bruss and B. K. P. Horn. Passive navigation. *Computer Vision, Graphics, and Image Processing*, 21:3–20, 1983.

[3] M. W. Fahle. Parallel, semi-parallel, and serial processing of visual hyperacuity. In *Proc. SPIE Conf. on Electronic Imaging: science and technology*, Santa Clara, CA, February 1990. to appear.

[4] E. C. Hildreth, N. M. Grzywacz, E. H. Adelson, and V. K. Inada. The perceptual buildup of three-dimensional structure from motion, 1989. Perception & Psychophysics, in press.

[5] J. J. Koenderink and A. J. van Doorn. Local structure of movement parallax of the plane. *Journal of the Optical Society of America*, 66:717–723, 1976.

[6] J. Moody and C. Darken. Fast learning in networks of locally tuned processing units. *Neural Computation*, 1:281–289, 1989.

[7] R. C. Nelson and J. Aloimonos. Using flow field divergence for obstacle avoidance: towards qualitative vision. In *Proceedings of the 2nd International Conference on Computer Vision*, pages 188–196, Tarpon Springs, FL, 1988. IEEE, Washington, DC.

[8] T. Poggio and F. Girosi. Regularization algorithms for learning that are equivalent to multilayer networks. *Science*, 247:978–982, 1990.

[9] T. Poggio and C. Koch. Ill–posed problems in early vision: from computational theory to analog networks. *Proceedings of the Royal Society of London B*, 226:303–323, 1985.

[10] R.Y. Tsai and T.S. Huang. Uniqueness and estimation of three dimensional motion parameters of rigid objects with curved surfaces. *IEEE Transactions on Pattern Analysis and Machine Intelligence*, 6:13–27, 1984.

[11] S. Ullman. Computational studies in the interpretation of structure and motion: summary and extension. In J. Beck, B. Hope, and A. Rosenfeld, editors, *Human and Machine Vision*. Academic Press, New York, 1983.

[12] S. Ullman. Maximizing rigidity: the incremental recovery of 3D structure from rigid and rubbery motion. *Perception*, 13:255–274, 1984.

[13] D. Weinshall. Direct computation of 3D shape and motion invariants. A.I. Memo No. 1131, Artificial Intelligence Laboratory, Massachusetts Institute of Technology, May 1989.
